# Learning temporally persistent hierarchical representations

**Suzanna Becker**
Department of Psychology
McMaster University
Hamilton, Ont. L8S 4K1
becker@mcmaster.ca

## Abstract

A biologically motivated model of cortical self-organization is proposed. Context is combined with bottom-up information via a maximum likelihood cost function. Clusters of one or more units are modulated by a common contextual gating signal; they thereby organize themselves into mutually supportive predictors of abstract contextual features. The model was tested in its ability to discover viewpoint-invariant classes on a set of real image sequences of centered, gradually rotating faces. It performed considerably better than supervised back-propagation at generalizing to novel views from a small number of training examples.

## 1 THE ROLE OF CONTEXT

The importance of context effects[1] in perception has been demonstrated in many domains. For example, letters are recognized more quickly and accurately in the context of words (see e.g. McClelland & Rumelhart, 1981), words are recognized more efficiently when preceded by related words (see e.g. Neely, 1991), individual speech utterances are more intelligible in the context of continuous speech, etc. Further, there is mounting evidence that neuronal responses are modulated by context. For example, even at the level of the LGN in the thalamus, the primary source of visual input to the cortex, Murphy & Sillito (1987) have reported cells with "end-stopped" or length-tuned receptive fields which depend on top-down inputs from the cortex. The end-stopped behavior disappears when the top-down connections are removed, suggesting that the cortico-thalamic connections are providing contextual modulation to the LGN. Moving a bit higher up the visual hierarchy, von der Heydt et al. (1984) found cells which respond to "illusory contours", in the absence of a contoured stimulus within the cells' classical receptive fields. These examples demonstrate that neuronal responses can be modulated by secondary sources of information in complex ways, provided the information is consistent with their expected or preferred input.

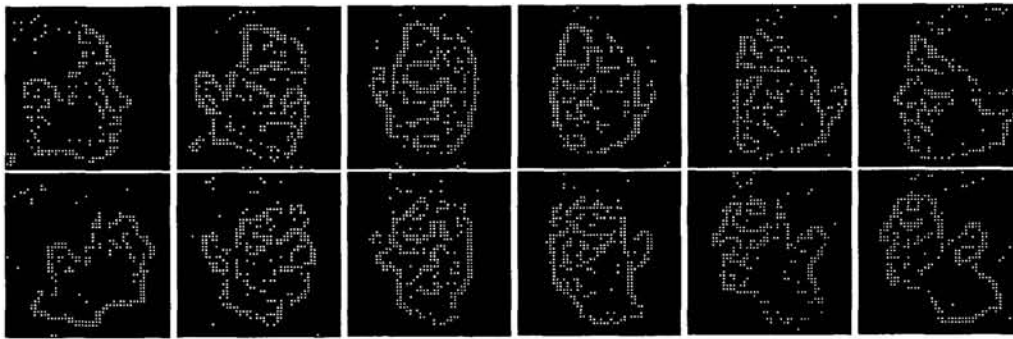

Figure 1: *Two sequences of 48 by 48 pixel images digitized with an IndyCam and prepro-cessed with a Sobel edge filter. Eleven views of each of four to ten faces were used in the simulations reported here. The alternate (odd) views of two of the faces are shown above.*

Why would contextual modulation be such a pervasive phenomenon? One obvious reason is that if context can influence *processing*, it can help in disambiguating or cleaning up a noisy stimulus. A less obvious reason may be that if context can influence *learning*, it may lead to more compact representations, and hence a more powerful processing system. To illustrate, consider the benefits of incorporating temporal history into an unsupervised classifier. Given a continuous sensory signal as input, the classifier must try to discover important partitions in its training data. If it can discover features that are *temporally persistent*, and thus insensitive to transformations in the input, it should be able to represent the signal compactly with a small set of features. Further, these features are more likely to be associated with the identity of objects rather than lower-level attributes.

However, most classifiers group patterns together on the basis of spatial overlap. This may be reasonable if there is very little shift or other form of distortion between one time step and the next, but is not a reasonable assumption about the sensory input to the cortex. Pre-cortical stages of sensory processing, certainly in the visual system (and probably in other modalities), tend to remove low-order correlations in space and time, e.g. with centre-surround filters. Consider the image sequences of gradually rotating faces in Figure 1. They have been preprocessed by a simple edge-filter, so that successive views of the same face have relatively little pixel overlap. In contrast, identical views of different faces may have considerable overlap. Thus, a classifier such as k-means, which groups patterns based on their Euclidean distance, would not be expected to do well at classifying these patterns. So how are people (and in fact very young children) able to learn to classify a virtually infinite number of objects based on relatively brief exposures? It is argued here that the assumption of *temporal persistence* is a powerful constraining factor for achieving this, and is one which may be used to advantage in artificial neural networks as well. Not only does it lead to the development of higher-order feature analyzers, but it can result in more compact codes which are important for applications like image compression. Further, as the simulations reported here show, improved generalization may be achieved by allowing high-level expectations (e.g. of class labels) to influence the development of lower-level feature detectors.

## 2   THE MODEL

Competitive learning (for a review, see Becker & Plumbley, 1996) is considered by many to be a reasonably strong candidate model of cortical learning. It can be implemented, in its simplest form, by a Hebbian learning rule in a network

with lateral inhibition. However, a major limitation of competitive learning, and the majority of unsupervised learning procedures (but see the Discussion section), is that they treat the input as a set of independent identically distributed (iid) samples. They fail to take into account context. So they are unable to take advantage of the temporal continuity in signals. In contrast, real sensory signals may be better viewed as discretely sampled, continuously varying time-series rather than iid samples.

The model described here extends maximum likelihood competitive learning (MLCL) (Nowlan, 1990) in two important ways: (i) modulation by context, and (ii) the incorporation of several "canonical features" of neocortical circuitry. The result is a powerful framework for modelling cortical self-organization.

MLCL retains the benefits of competitive learning mentioned above. Additionally, it is more easily extensible because it maximizes a global cost function:

$$L = \sum_{\alpha=1}^{n} \log \left[ \sum_{i=1}^{m} \pi_i y_i^{(\alpha)} \right] \tag{1}$$

where the $\pi_i$'s are positive weighting coefficients which sum to one, and the $y_i$'s are the clustering unit activations:

$$y_i^{(\alpha)} = N(\vec{I}^{(\alpha)}, \vec{w}_i, \Sigma_i) \tag{2}$$

where $\vec{I}^{(\alpha)}$ is the input vector for pattern $\alpha$, and $N()$ is the probability of $\vec{I}^{(\alpha)}$ under a Gaussian centred on the $i$th unit's weight vector, $\vec{w}_i$, with covariance matrix $\Sigma_i$. For simplicity, Nowlan used a single global variance parameter for all input dimensions, and allowed it to shrink during learning. MLCL actually maximizes the log likelihood $(L)$ of the data under a mixture of Gaussians model, with mixing proportions equal to the $\pi$'s. $L$ can be maximized by online gradient ascent[2] with learning rate $\varepsilon$:

$$\Delta w_{ij} = \varepsilon \; \frac{\partial L}{\partial w_{ij}} = \varepsilon \sum_{\alpha} \frac{\pi_i \; y_i^{(\alpha)}}{\sum_k \pi_k \; y_k^{(\alpha)}} \left( I_j^{(\alpha)} - w_{ij} \right) \tag{3}$$

Thus, we have a Hebbian update rule with normalization of post-synaptic unit activations (which could be accomplished by shunting inhibition) and weight decay.

## 2.1 Contextual modulation

To integrate a contextual information source into MLCL, our first extension is to replace the mixing proportions ($\pi_i$'s) by the outputs of *contextual gating units* (see Figure 2). Now the $\pi_i$'s are computed by separate processing units receiving their own separate stream of input, the "context". The role of the gating signals here is analagous to that of the gating network in the (supervised) "competing experts" model (Jacobs et al., 1991).[3] For the network shown in Figure 2, the context is simply a time-delayed version of the outputs of a *module* (explained in the next subsection). However, more general forms of context are possible (see Discussion). In the simulations reported here, the context units computed their outputs according to a *softmax* function of their weighted summed inputs $x_i$:

$$\pi_i^{(\alpha)} = \frac{e^{x_i(\alpha)}}{\sum_j e^{x_j(\alpha)}} \tag{4}$$

We refer to the action of the gating units (the $\pi_i$'s) as *modulatory* because of the

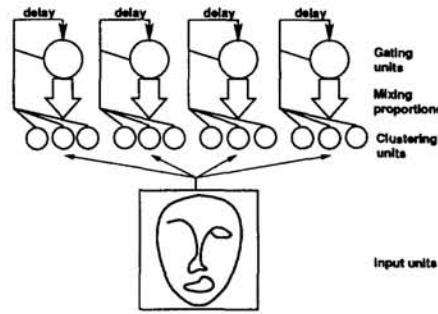

Figure 2: *The architecture used in the simulations reported here. Except where indicated, the gating units received all their inputs across unit delay lines with fixed weights of 1.0.*

multiplicative effect they have on the activities of the clustering units (the $y_i$'s). This multiplicative interaction is built into the cost function (Equation 1), and consequently, arises in the learning rule (Equation 3). Thus, clustering units are encouraged to discover features that agree with the current context signal they receive. If their context signal is weak or if they fail to capture enough of the activation relative to the other clustering units, they will do very little learning. Only if a unit's weight vector is sufficiently close to the current input vector *and* it's corresponding gating unit is strongly active will it do substantial learning.

## 2.2 Modular, hierarchical architecture

Our second modification to MLCL is required to apply it to the architecture shown in Figure 2, which is motivated by several ubiquitous features of the neocortex: a laminar structure, and a functional organization into "cortical clusters" of spatially nearby columns with similar receptive field properties (see e.g. Calvin, 1995). The cortex, when flattened out, is like a large six-layered sheet. As Calvin (1995, pp. 269) succinctly puts it, "... the bottom layers are like a subcortical 'out' box, the middle layer like an 'in' box, and the superficial layers somewhat like an 'interoffice' box connecting the columns and different cortical areas". The middle and superficial layer cells are analagous to the first-layer clustering units and gating units respectively. Thus, we propose that the superficial cells may be providing the contextual modulation. (The bottom layers are mainly involved in motor output and are not included in the present model.) To induce a functional modularity in our model analogous to cortical clusters, clustering units within the same module receive a *shared gating signal*. The cost function and learning rule are now:

$$L = \sum_{\alpha=1}^{n} \log \left[ \sum_{i=1}^{m} \pi_i^{(\alpha)} \frac{1}{l} \sum_{j=1}^{l} y_{ij}^{(\alpha)} \right] \tag{5}$$

$$\Delta w_{ijk} = \varepsilon \sum_{\alpha} \frac{\pi_i^{(\alpha)} y_{ij}^{(\alpha)}}{\sum_q \pi_q^{(\alpha)} \sum_r y_{qr}^{(\alpha)}} \left( I_k^{(\alpha)} - w_{ijk} \right) \tag{6}$$

Thus, units in the same module form predictions $y_{ij}^{(\alpha)}$ of the same contextual feature $\pi_i^{(\alpha)}$. Fortunately, there is a disincentive to all of them discovering identical weights: they would then do poorly at modelling the input.

## 3 EXPERIMENTS

As a simple test of this model, it was first applied to a set of image sequences of four centered, gradually rotating faces (see Figure 1), divided into training and test

|                      |         | Training Set | Test Set   |
|----------------------|---------|--------------|------------|
| no context, 4 faces: | Layer 1 | 59.2 (2.4)   | 65 (3.5)   |
| context, 4 faces:    | Layer 1 | 88.4 (3.9)   | 74.5 (4.2) |
|                      | Layer 2 | 88.8 (4.0)   | 72.7 (4.8) |
| context, 10 faces:   | Layer 1 | 96.3 (1.2)   | 71.0 (3.0) |
|                      | Layer 2 | 91.8 (2.4)   | 70.2 (4.3) |

Table 1: *Mean percent (and standard error) correctly classified faces, across 10 runs, for unsupervised clustering networks trained for 2000 iterations with a learning rate of 0.5, with and without temporal context. Layer 1: clustering units. Layer 2: gating units. Performance was assessed as follows: Each unit was assigned to predict the face class for which it most frequently won (was the most active). Then for each pattern, the layer's activity vector was counted as correct if the winner correctly predicted the face identity.*

sets by taking alternating views. It was predicted that the clustering units should discover "features" such as individual views of specific faces. Further, different views of the same face should be clustered together within a module because they will be observed in the same temporal context, while the gating units should discover the identity of faces, independent of viewpoint.

First, the baseline effect of the temporal context on clustering performance was assessed by comparing the network shown in Figure 2 to the same network with the input connections to the gating layer removed. The latter is equivalent to MLCL with fixed, equal $\pi_i$'s. The results are summarized in Table 1. As predicted, the temporal context provides incentive for the clustering units to group successive instances of the same face together, and the gating layer can therefore do very well at classifying the faces with a much smaller number of units - i.e., independently of viewpoint. In contrast, the clustering units without the contextual signal are more likely to group together similar views of different people's faces.

Next, to explore the scaling properties of the model, a network like the one shown in Figure 2 but with 10 modules was presented with a set of 10 faces, 11 views each. As before, the odd-numbered views were trained on and the even-numbered views were tested on. To achieve comparable performance to the smaller network, the weights on the self-pointing connections on the gating units were increased from 1.0 to 3.0, which increased the time constant of temporal agveraging. The model then had no difficulty scaling up to the larger training set size, as shown in Table 1.

Based on the unexpected success of this model, it's classification performance was then compared against supervised back-propagation networks on the four face sequences. The first supervised network we tried was a simple recurrent network with essentially the same architecture: one layer of Gaussian units followed by one layer of recurrent softmax units with fixed delay lines. Over ten runs of each model, although the unsupervised classifier did worse on the training set (it averaged 88% while the supervised model always scored 100% correct), it outperformed the supervised model in its generalization ability by a considerable margin (it averaged 73% while the supervised model averaged 45% correct).

Finally, a feedforward back-propagation network with sigmoid units was trained. The following constraint on the hidden layer activations, $h_j(t)$: [4]

$$\text{hidden state cost} = \lambda \sum_j (h_j(t) - h_j(t-1))^2$$

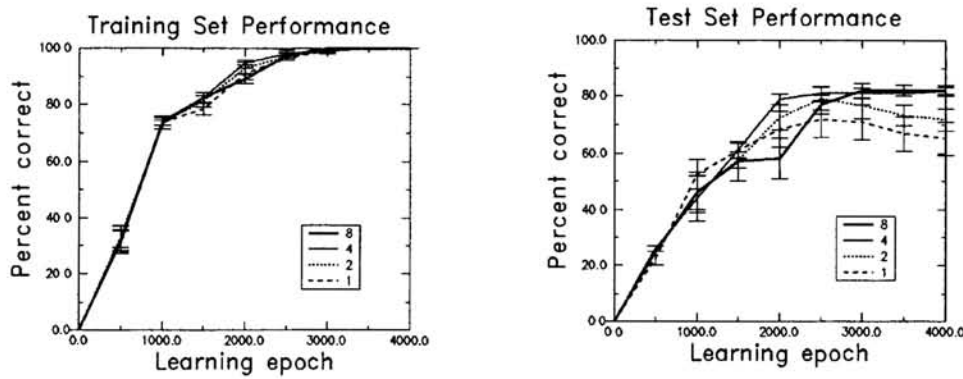

Figure 3: *Learning curves, averaged over five runs, for feedforward supervised net with a temporal smoothness constraint, for each of four levels of the parameter* $\lambda$.

was added to the cost function to encourage temporal smoothness. As the results in Figure 3 show, a feedforward network with no contextual input was thereby able to perform as well as our unsupervised model when it was constrained to develop hidden layer representations that clustered temporally adjacent patterns together.

## 4   DISCUSSION

The unsupervised model's markedly better ability to generalize stems from it's cost function; it favors hidden layer features which contribute to temporally coherent predictions at the output (gating) layer. Multiple views of a given object are therefore more likely to be detected by a given clustering unit in the unsupervised model, leading to considerably improved interpolation of novel views. The poor generalization performance of back-propagation is not just due to overtraining, as the learning curves in Figure 3 show. Even with early stopping, the network with the lowest value of $\lambda$ would not have done as well as the unsupervised network. There is simply no reason why supervised back-propagation should cluster temporally adjacent views together unless it is explicitly forced to do so.

A "contextual input" stream was implemented in the simplest possible way in the simulations reported here, using fixed delay lines. However, the model we have proposed provides for a completely general way of incorporating arbitrary contextual information, and could equally well integrate other sources of input. The incoming weights to the gating units could also be learned. In fact, the gating unit activities actually represent the probabilities of each clustering unit's Gaussian model fitting the data, conditioned on the temporal history; hence, the entire model could be viewed as a Hidden Markov Model (Geoff Hinton, personal communication). However, current techniques for fitting HMMs are intractable if state dependencies span arbitrarily long time intervals.

The model in its present implementation is not meant to be a realistic account of the way humans learn to recognize faces. Viewpoint-invariant recognition is achieved, if at all, in a hierarchical, multi-stage system. One could easily extend our model to achieve this, by connecting together a sequence of networks like the one shown in Figure 2, each having progressively larger receptive fields.

A number of other unsupervised learning rules have been proposed based on the assumption of temporally coherent inputs (Földiák, 1991; Becker, 1993; Stone, 1996). Phillips et al. (1995) have proposed an alternative model of cortical self-organization they call *coherent Infomax* which incorporates contextual modulation. In their model, the outputs from one processing stream modulate the activity in another

stream, while the mutual information between the two streams is maximized.

A wide range of perceptual and cognitive abilities could be modelled by a network that can learn features of its primary input in particular contexts. These include multi-sensor fusion, feature segregation in object recognition using top-down cues, and semantic disambiguation in natural language understanding. Finally, it is widely believed that memories are stored rapidly in the hippocampus and related brain structures, and gradually incorporated into the slower-learning cortex for long-term storage. The model proposed here may be able to explain how such interactions between disparate information sources are learned.

## Acknowledgements

This work evolved out of discussions with Ron Racine and Larry Roberts. Thanks to Geoff Hinton for contributing several valuable insights, as mentioned in the paper, and to Ken Seergobin for the face images. Software was developed using the Xerion neural network simulation package from Hinton's lab, with programming assistance from Lianxiang Wang. This work was supported by a McDonnell-Pew Cognitive Neuroscience research grant and a research grant from the Natural Sciences and Engineering Research Council of Canada.

## Footnotes

[1]We use the term context rather loosely here to mean any secondary source of input. It could be from a different sensory modality, a different input channel within the same modality, a temporal history of the input, or top-down information.

[2]Nowlan (1990) used a slightly different online weight update rule that more closely approximates the batch update rule of the EM algorithm (Dempster et al., 1977)

[3]However, in the competing experts architecture, both the experts and gating network receive a common source of input. The competing experts model could be thought of as fitting a mixture model of the training signal.

[4]As Geoff Hinton pointed out, the above constraint, if normalized by the variance, maximizes the mutual information between hidden unit states at adjacent time steps.

## References

Becker, S. (1993). Learning to categorize objects using temporal coherence. In S. J. Hanson, J. D. Cowan, & C. L. Giles (Eds.), *Advances in Neural Information Processing Systems 5* (pp. 361–368). San Mateo, CA: Morgan Kaufmann.

Becker, S. & Plumbley, M. (1996). Unsupervised neural network learning procedures for feature extraction and classification. *International Journal of Applied Intelligence, 6*(3).

Calvin, W. H. (1995). Cortical columns, modules, and Hebbian cell assemblies. In M. Arbib (Ed.), *The handbook of brain theory and neural networks*. Cambridge, MA: MIT Press.

Dempster, A. P., Laird, N. M., & Rubin, D. B. (1977). Maximum likelihood from incomplete data via the EM algorithm. *Proceedings of the Royal Statistical Society, B-39*:1–38.

Földiák, P. (1991). Learning invariance from transformation sequences. *Neural Computation, 3*(2):194–200.

Jacobs, R. A., Jordan, M. I., Nowlan, S. J., & Hinton, G. E. (1991). Adaptive mixtures of local experts. *Neural Computation, 3*(1):79–87.

McClelland, J. L. & Rumelhart, D. E. (1981). An interactive activation model of context effects in letter perception, part I: An account of basic findings. *Psychological Review, 88*:375–407.

Murphy, C. & Sillito, A. M. (1987). Corticofugal feedback influences the generation of length tuning in the visual pathway. *Nature, 329*:727–729.

Neely, J. (1991). Semantic priming effects in visual word recognition: A selective review of current findings and theories. In D. Besner & G. W. Humphreys (Eds.), *Basic processes in reading: Visual Word Recognition* (pp. 264–336). Hillsdale, NJ: Lawrence Erlbaum Associates.

Nowlan, S. J. (1990). Maximum likelihood competitive learning. In D. S. Touretzky (Ed.), *Neural Information Processing Systems, Vol. 2* (pp. 574–582). San Mateo, CA: Morgan Kaufmann.

Phillips, W. A., Kay, J., & Smyth, D. (1995). The discovery of structure by multi-stream networks of local processors with contextual guidance. *Network, 6*:225–246.

Stone, J. (1996). Learning perceptually salient visual parameters using spatiotemporal smoothness constraints. *Neural Computation, 8*:1463–1492.

von der Heydt, R., Peterhans, E., & Baumgartner, G. (1984). Illusory contours and cortical neural responses. *Science, 224*:1260–1262.